# On-Line Estimation of the Optimal Value Function: HJB-Estimators

**James K. Peterson**
Department of Mathematical Sciences
Martin Hall Box 341907
Clemson University
Clemson, SC 29634-1907
email: `peterson@math.clemson.edu`

## Abstract

In this paper, we discuss on-line estimation strategies that model the optimal value function of a typical optimal control problem. We present a general strategy that uses local corridor solutions obtained via dynamic programming to provide *local* optimal control sequence training data for a neural architecture model of the optimal value function.

## 1   ON-LINE ESTIMATORS

In this paper, the problems of adaptive control using neural architectures are explored in the setting of general on-line estimators. We will try to pay close attention to the underlying mathematical structure that arises in the on-line estimation process.

The complete effect of a control action $u_k$ at a given time step $t_k$ is clouded by the fact that the state history depends on the control actions taken after time step $t_k$. So the effect of a control action over *all future time* must be monitored. Hence, choice of control must inevitably involve knowledge of the future history of the state trajectory. In other words, the optimal control sequence can not be determined until after the fact. Of course, standard optimal control theory supplies an optimal control sequence to this problem for a variety of performance criteria. Roughly, there are two approaches of interest: solving the two-point boundary value

problem arising from the solution of Pontryagin's maximum or minimum principle or solving the Hamilton-Jacobi-Bellman (HJB) partial differential equation. However, the computational burdens associated with these schemes may be too high for real-time use. Is it possible to essentially use on-line estimation to build a solution to either of these two classical techniques at a lower cost? In other words, if $\eta$ samples are taken of the system from some initial point under some initial sequence of control actions, can this time series be use to obtain information about the true optimal sequence of controls that should be used in the next $\eta$ time steps?

We will focus here on algorithm designs for on-line estimation of the optimal control law that are implementable in a control step time of 20 milliseconds or less. We will use local learning methods such as CMAC (Cerebellar Model Articulated Controllers) architectures (Albus, 1 and W. Miller, 7), and estimators for characterizations of the optimal value function via solutions of the Hamilton-Jacobi-Bellman equation, (adaptive critic type methods), (Barto, 2; Werbos, 12).

## 2   CLASSICAL CONTROL STRATEGIES

In order to discuss on-line estimation schemes based on the Hamilton- Jacobi-Bellman equation, we now introduce a common sample problem:

$$\min_{u \in \mathcal{U}} \quad \hat{J}(x, u, t) \tag{1}$$

where

$$\hat{J}(x, u, t) = dist(y(t_f), \Gamma) + \int_t^{t_f} L(y(s), u(s), s)\, ds \tag{2}$$

Subject to:

$$y'(s) = f(y(s), u(s), s),\ t \leq s \leq t_f \tag{3}$$
$$y(t) = x \tag{4}$$
$$y(s) \in \mathcal{Y}(s) \subseteq R^N,\ t \leq s \leq t_f \tag{5}$$
$$u(s) \in U(s) \subseteq R^M,\ t \leq s \leq t_f \tag{6}$$

Here $y$ and $u$ are the *state vector* and *control vector* of the system, respectively; $\mathcal{U}$ is the space of functions that the control must be chosen from during the minimization process and ( 4) - ( 6) give the initialization and constraint conditions that the state and control must satisfy. The set $\Gamma$ represents a target constraint set and $dist(y(t_f), \Gamma)$ indicates the distance from the final state $y(t_f)$ to the constraint set $\Gamma$. The optimal value of this problem for the initial state $x$ and time $t$ will be denoted by $J(x, t)$ where

$$J(x, t) = \min_u \hat{J}(x, u, t).$$

It is well known that the optimal value function $J(x,t)$ satisfies a generalized partial differential equation known as the Hamilton-Jacobi-Bellman (HJB) equation.

$$-\frac{\partial J(x,t)}{\partial t} = \min_u \left\{ L(x,u,t) + \frac{\partial J(x,t)}{\partial x} f(x,u,t) \right\}$$
$$J(x,t_f) = dist(x,\Gamma)$$

In the case that $J$ is indeed differentiable with respect to both the state and time arguments, this equation is interpreted in the usual way. However, there are many problems where the optimal value function is not differentiable, even though it is bounded and continuous. In these cases, the optimal value function $J$ can be interpreted as a *viscosity solution* of the HJB equation and the partial derivatives of $J$ are replaced by the *sub* and *superdifferentials* of $J$ (Crandall, 5). In general, once the HJB equation is solved, the optimal control from state $x$ and time $t$ is then given by the minimum condition

$$u \in arg\min_u \left\{ L(x,u,t) + \frac{\partial J(x,t)}{\partial x} f(x,u,t) \right\}$$

If the underlying state and time space are discretized using a state mesh of resolution $r$ and a time mesh of resolution $s$, the HJB equation can be rewritten into the form of the standard Bellman Principle of Optimality (BPO):

$$J_{rs}(x_i,t_j) = \min_u \{ L(x_i,u,t_j)(t_{j+1} - t_j) + J_{rs}(x(x_i,u),t_{j+1}) \}$$

where $x(x_i,u)$ indicates the new state achieved by using control $u$ over time interval $[t_j,t_{j+1}]$ from initial state $x_i$. In practice, this equation is solved by successive iterations of the form:

$$J_{rs}^{\tau+1}(x_i,t_j) = \min_u \{ L(x_i,u,t_j)(t_{j+1} - t_j) + J_{rs}^{\tau}(x(x_i,u),t_{j+1}) \}$$

where $\tau$ denotes the iteration cycle and the process is started by initializing $J_{rs}^0(x_i,t_j)$ in a suitable manner. Generally, the iterations continue until the values $J_{rs}^{\tau+1}(x_i,t_j)$ and $J_{rs}^{\tau+1}(x_i,t_j)$ differ by negligible amounts. This iterative process is usually referred to as *dynamic programming* (DP). Once this iterative process converges, let $J_{rs}(x_i,t_j) = \lim_{\tau\to\infty} J_{rs}^{\tau}$, and consider $\lim_{(r,s)\to(0,0)} J_{rs}(x_i^r,t_j^s)$, where $(x_i^r,t_j^s)$ indicates that the discrete grid points depend on the resolution $(r,s)$. In many situations, this limit gives the viscosity solution $J(x,t)$ to the HJB equation.

Now consider the problem of finding $J(x,0)$. The Pontryagin minimum principle gives first order necessary conditions that the optimal state $x$ and costate $p$ variables must satisfy. Letting $H(x,u,p,t) = L(x,u,t) + p^T f(x,u,t)$ and defining

$$H(x,p,t) \quad = \quad \min_{u} \hat{H}(x,u,p,t), \tag{7}$$

the optimal state and costate then must satisfy the following two-point boundary value problem (TPBVP):

$$
\begin{array}{ll}
x'(t) = \frac{\partial H(x,p,t)}{\partial p}, & p'(t) = - \frac{\partial H(x,p,t)}{\partial x} \\
x(0) = x, & p(t_f) = 0
\end{array}
\tag{8}
$$

and the optimal control is obtained from ( 7) once the optimal state and costate are determined. Note that ( 7) can not necessarily be solved for the control $u$ in terms of $x$ and $p$, i.e. a feedback law may not be possible. If the TPBVP can not be solved, then we set $J(x,0) = \infty$. In conclusion, in this problem, we are led inevitably to an optimal value function that can be poorly behaved; hence, we can easily imagine that at many $(x,t)$, $\frac{\partial J}{\partial x}$ is not available and hence $J$ will not satisfy the HJB equation in the usual sense. So if we estimate $J$ directly using some form of on-line estimation, how can we hope to back out the control law if $\frac{\partial J}{\partial x}$ is not available?

## 3    HJB ESTIMATORS

A potential on-line estimation technique can be based on approximations of the optimal value function. Since the optimal value function should satisfy the HJB equation, these methods will be grouped under the broad classification **HJB estimators**.

Assume that there is a given initial state $x_0$ with start time 0. Consider a local patch, or *local corridor*, of the state space around the initial state $x_0$, denoted by $\Omega(x_0)$. The exact size of $\Omega(x_0)$ will depend on the nature of the state dynamics and the starting state. If $\Omega(x_0)$ is then discretized using a coarse grid of resolution $r$ and the time domain is discretized using resolution $s$, an approximate dynamic programming problem can be formulated and solved using the BPO equations. Since the new states obtained via integration of the plant dynamics will in general not land on coarse grid lines, some sort of interpolation must be used to assign the integrated new state value an appropriate coarse grid value. This can be done using the coarse encoding implied by the grid resolution $r$ of $\Omega(x_0)$. In addition, multiple grid resolutions may be used with coarse and fine grid approximations interacting with one another as in multigrid schemes (Briggs, 3). The optimal value function so obtained will be denoted by $J_{rs}(z_i, t_j)$ for any discrete grid point $z_i \in \Omega(x_0)$ and time point $t_j$. This approximate solution also supplies an estimate of the optimal control sequence $(u^*)_{ij}^{\eta-1} \equiv (u^*)_j^{\eta-1}(z_i, t_j)$. Some papers on approximate dynamic programming are (Peterson, 8; (Sutton, 10; Luus, 6). It is also possible to obtain estimates of the optimal control sequences, states and costates using an $\eta$ step look-ahead and the Pontryagin minimum principle. The associated two point boundary value problem is solved and the controls computed via $u_i \in arg \min_u \hat{H}(x_i^*, u, p_i^*, t_i)$ where $(x^*)_0^\eta$ and $(p^*)_0^\eta$ are the calculated optimal state and costate sequences respectively. This approach is developed in (Peterson, 9) and implemented for

vibration suppression in a large space structure, by (Carlson, Rothermel and Lee, 4)

For any $z_i \in \Omega(x_0)$, let $(u)_{ij}^{\eta-1} \equiv (u)_j^{\eta-1}(z_i, t_j)$ be a control sequence used from initial state $z_i$ and time point $t_j$. Thus $u_{ij}$ is the control used on time interval $[t_j, t_{j+1}]$ from start point $z_i$. Define $z_{ij}^{j+1} \equiv z(z_i, u_{ij}, t_j)$, the state obtained by integrating the plant dynamics one time step using control $u_{ij}$ and initial state $z_i$. Then $u_{i,j+1}$ is the control used on time interval $[t_{j+1}, t_{j+2}]$ from start point $z_{ij}^{j+1}$ and the new state is $z_{ij}^{j+2} \equiv z(z_{ij}^{j+1}, u_{i,j+1}, t_{j+1})$; in general, $u_{i,j+k}$ is the control used on time interval $[t_{j+k}, t_{j+k+1}]$ from start point $z_{ij}^{j+k}$ and the new state is $z_{ij}^{j+k+1} \equiv z(z_{ij}^{j+k}, u_{i,j+k}, t_{j+k})$, where $z_{ij}^{j} \equiv z_i$.

Let's now assume that optimal control information $u_{ij}$ (we will dispense with the superscript $*$ labeling for expositional cleanness) is available at each of the discrete grid points $(z_i, t_j) \in \Omega(x_0)$. Let $\phi_{rs}(z_i, t_j)$ denote the value of a neural architecture (CMAC, feedforward, associative etc.) which is trained as follows using this optimal information for $0 \le k < \eta - j - 1$ (the equation below holds for the converged value of the network's parameters and the actual dependence of the network on those parameters is notationally suppressed):

$$\phi_{rs}(z_{ij}^{j+k}, t_{j+k}) = \xi \phi_{rs}(z_{ij}^{j+k+1}, t_{j+k+1}) + \zeta \Re(z_{ij}^{j+k}, u_{i,j+k}) \tag{9}$$

where $0 < \xi, \zeta \le 1$ and we define a typical reinforcement function $\Re$ by

$$\Re(z_{ij}^{j+k}, u_{i,j+k}, t_{j+k}, t_{j+k+1}) = \tag{10}$$

$$\begin{cases} L(z_{ij}^{j+k}, u_{i,j+k}, t_{j+k})(t_{j+k+1} - t_{j+k}) & \text{if } j \le k < \eta - j - 1 \\ L(z_{ij}^{\eta-1}, u_{i,\eta-1}, t_{\eta-1})(t_\eta - t_{\eta-1}) & \text{if } k = \eta - 1 \\ \quad + dist(z_{ij}^\eta, \Gamma) \end{cases} \tag{11}$$

For notational convenience, we will now drop the notational dependence on the time grid points and simply refer to the reinforcement by $\Re(z_{ij}^{j+k}, u_{i,j+k})$

Then applying ( 9) repeatedly, for any $0 \le p \le \eta - i$,

$$\phi_{rs}(z_i, t_j) = \xi^p \phi_{rs}(z_{ij}^{j+p}, t_{j+p}) + \zeta \sum_{k=0}^{p-1} \xi^k \Re(z_{ij}^{j+k}, u_{i,j+k}) \tag{12}$$

Thus, the function $\Psi_{rs}$ can be defined by

$$\begin{aligned} \Psi_{rs}(z_i, t_j, \xi, \zeta) &= (\zeta)^{-1} \phi_{rs}(z_i, t_j) - \xi^p \phi_{rs}(z_{ij}^\eta, t_\eta) \\ &= \sum_{k=0}^{\eta-j-1} \xi^k \Re(z_{ij}^{j+k}, u_{i,j+k}), \end{aligned}$$

where the term $u_{j\eta}$ will be interpreted as $u_{j,\eta-1}$.

It follows then that since $u_{ij}$ is optimal,

$$\Psi_{rs}(z_i, t_j, 1, 1) \quad = \quad J_{rs}(z_i, t_j)$$

Clearly, the function $\Phi_{rs}(z_i, t_j) = \Psi_{rs}(z_i, t_j, 1, 1)$ estimates the optimal value $J_{rs}(z_i, t_j)$ itself. (See, Q-Learning (Watkins, 11)).

An alternate approach that does not model $J$ indirectly, as is done above, is to train a neural model $\Phi_{rs}(z_i, t_j)$ directly on the data $J(z_i, t_j)$ that is computed in each local corridor calculation. In either case, the above observations lead to the following algorithm:

**Initialization:**
Here, the iteration count is $\tau = 0$. For given starting state $x_0$ and local look ahead of $\eta$ time steps, form the local corridor $\Omega(x_0)$ and solve the associated approximate BPO equation for $J_{rs}(z_i, t_j)$. Compute the associated optimal control sequences for each $(z_i, t_j)$ pair, $(u^*)_{ij}^{\eta-1} \equiv (u^*)_{j}^{\eta-1}(z_i, t_j)$. Initialize the neural architecture for the optimal value estimate using $\Phi_{rs}^0(z_i, t_j) = J_{rs}(z_i, t_j)$.

**Estimate of New Optimal Control Sequence:**
For the next $\eta$ time steps, an estimate must be made of the next optimal control action in time interval $[t_{\eta+k}, t_{\eta+k+1}]$. The initial state is any $z_i$ in $\Omega(x_\eta)$ ($x_\eta$ is one such choice) and the initial time is $t_\eta$. For the time interval $[t_\eta, t_{\eta+1}]$, if the model $\Phi_{rs}^0(z_i, t_j)$ is differentiable, the new control can be estimated by

$$\hat{u}_{\eta+1} \quad \in \quad arg \min_{u} \left\{ \begin{array}{l} L(z_\eta, u, t_\eta)(t_{\eta+1} - t_\eta) \\ +\frac{\partial \Phi_{rs}^0}{\partial x}(z_\eta, t_\eta) \\ f(z_\eta, u, t_\eta)(t_{\eta+1} - t_\eta) \end{array} \right\}$$

For ease of notation, let $z_{\eta+1}$ denote the new state obtained using the control $u_{\eta+1}$ on the interval $[t_\eta, t_{\eta+1}]$. Then choose the next control via

$$\hat{u}_{\eta+2} \quad \in \quad arg \min_{u} \left\{ \begin{array}{l} L(z_{\eta+1}, u, t_{\eta+1})(t_{\eta+2} - t_{\eta+1}) \\ +\frac{\partial \Phi_{rs}^0}{\partial x}(z_{\eta+1}, t_{\eta+1}) \\ f(z_{\eta+1}, u, t_{\eta+1})(t_{\eta+2} - t_{\eta+1}) \end{array} \right\}$$

Clearly, if $z_{\eta+k}$ denote the new state obtained using the control $u_{\eta+k-1}$ on the interval $[t_{\eta+k}, t_{\eta+k+1}]$, the next control is chosen to satisfy

$$\hat{u}_{\eta+k} \quad \in \quad arg \min_{u} \left\{ \begin{array}{l} L(z_{\eta+k}, u, t_{\eta+k})(t_{\eta+k+1} - t_{\eta+k}) \\ +\frac{\partial \Phi_{rs}^0}{\partial x}(z_{\eta+k}, t_{\eta+k}) \\ f(z_{\eta+k}, u, t_{\eta+k})(t_{\eta+k+1} - t_{\eta+k}) \end{array} \right\}$$

Alternately, if the neural architecture is not differentiable (that is $\frac{\partial \Phi_{rs}^0}{\partial x}$ is not available), the new control action can be computed via

$$\hat{u}_{\eta+k} \quad \in \quad arg \min_{u} \left\{ \begin{array}{l} L(z_{\eta+k}, u, t_{\eta+k})(t_{\eta+k+1} - t_{\eta+k}) \\ + \Phi_{rs}^0(z_{\eta+k}(u), t_{\eta+k+1}) \end{array} \right\}.$$

**Update of the Neural Estimator:**

The new starting point for the dynamics is now $x_\eta$ and there is a new associated local corridor $\Omega(x_\eta)$. The neural estimator is then updated using either the HJB or the BPO equations over the local corridor $\Omega(x_\eta)$. Using the BPO equations, for all $z_i \in \Omega(x_\eta)$ the updates are:

$$\Phi_{rs}^1(z_i, t_{\eta+j}) \quad = \quad \min_{u}\{L(z_i, u, t_{\eta+j})(t_{\eta+j+1} - t_{\eta+j}) + \Phi_{rs}^0(z_i, t_{\eta+j})\}$$

where $(\hat{u})_j^{\eta-1}$ indicates the optimal control estimates obtained in the previous algorithm step. Finally, using the HJB equation, for all $z_i \in \Omega(x_\eta)$ the updates are:

$$\Phi_{rs}^1(z_i, t_{\eta+j}) \quad = \quad \Phi_{rs}^0(z_i, t_{\eta+j+1}) + \min_{u} \left\{ \begin{array}{l} L(z_i, u, t_{\eta+j})(t_{\eta+j+1} - t_{\eta+j}) \\ + \frac{\partial \Phi_{rs}^0}{\partial x}(z_i, t_{\eta+j}) \\ f(z_i, u, t_{\eta+j})(t_{\eta+j+1} - t_{\eta+j}) \end{array} \right\}$$

**Comparison to BPO optimal control sequence:**

Now solve the associated approximate BPO equation for each $z_i$ in the local corridor $\Omega(x_\eta)$ for $J_{rs}(z_i, t_{\eta+j})$. Compute the new approximate optimal control sequences for each $(z_i, t_{\eta+j})$ pair, $(u^*)_{\eta+j}^{2\eta-1} \equiv (u^*)_{\eta+j}^{2\eta-1}(z_i, t_{\eta+j})$ and compare them to the estimated sequences $(\hat{u})_{\eta+j}^{2\eta-1}$. If the discrepancy is out of tolerance (this is a design decision) initialize the neural architecture for the optimal value estimate using $\Phi_{rs}^1(z_i, t_{\eta+i}) = J_{rs}(z_i, t_{\eta+j})$. If the discrepancy is acceptable, terminate the BPO approximation calculations for $M$ future iterations and use the neural architectures alone for on-line estimation.

The determination of the stability and convergence properties of any on-line approximation procedure of this sort is intimately connected with the the optimal value function which solves the generalized HJB equation. We conjecture the following limit converges to a viscosity solution of the HJB equation for the given optimal control problem:

$$lim_{(r,s)\to(0,0)} lim_{\tau\to\infty} \Phi_{rs}^\tau(x_i^r, t_j^s) \quad = \quad J(x, t)$$

Further, there are stability questions and there are interesting issues relating to the use of multiple state resolutions $r_1$ and $r_2$ and the corresponding different approximations to $J$, leading to the use of multigrid like methods on the HJB equation (see, for example, Briggs, 3). Also note that there is an advantage to using CMAC

architectures for the approximation of the optimal value function $J$; since $J$ need not be smooth, the CMAC's lack of differentiability with respect to its inputs is not a problem and in fact is a virtue.

**Acknowledgements**

We acknowledge the partial support of NASA grant NAG 3-1311 from the Lewis Research Center.

**References**

1. Albus, J. 1975. "A New Approach to Manipulator Control: The Cerebellar Model Articulation Controller (CMAC)." *J. Dynamic Systems, Measurement and Control*, 220 - 227.

2. Barto, A., R. Sutton, C. Anderson. 1983 "Neuronlike Adaptive Elements That Can Solve Difficult Learning Control Problems." *IEEE Trans. Systems, Man Cybernetics*, Vol. SMC-13, No. 5, September/October, 834 - 846.

3. Briggs, W. 1987. **A Multigrid Tutorial**, SIAM, Philadelphia, PA.

4. Carlson, R., C. Lee and K. Rothermel. 1992. "Real Time Neural Control of an Active Structure", **Artificial Neural Networks in Engineering 2**, 623 - 628.

5. Crandall, M. and P. Lions. 1983. "Viscosity solutions of Hamilton-Jacobi Equations." *Trans. American Math. Soc.*, Vol. 277, No. 1, 1 - 42.

6. Luus, R. 1990. "Optimal Control by Dynamic Programming Using Systematic Reduction of Grid Size", *Int. J. Control*, Vol. 51, No. 5, 995 - 1013.

7. Miller, W. 1987. "Sensor-Based Control of Robotic Manipulators Using as General Learning Algorithm." *IEEE J. Robot. Automat.*, Vol RA-3, No. 2, 157 - 165

8. Peterson, J. 1992. "Neural Network Approaches to Estimating Directional Cost Information and Path Planning in Analog Valued Obstacle Fields", *HEURISTICS: The Journal of Knowledge Engineering*, Special Issue on Artificial Neural Networks, Vol. 5, No. 2, Summer, 50 - 61.

9. Peterson, J. 1992. "On-Line Estimation of Optimal Control Sequences: Pontryagin Estimators", **Artificial Neural Networks in Engineering 2**, ed. Dagli et. al., 579 - 584.

10. Sutton, R. 1991. "Planning by Incremental Dynamic Programming", *Proceedings of the Ninth International Workshop on Machine Learning*, 353 - 357.

11. Watkins, C. 1989. **Learning From Delayed Rewards**, Ph. D. Dissertation, King's College.

12. Werbos, P. 1990. "A Menu of Designs for Reinforcement Learning Over Time". In **Neural Networks for Control**, Ed. Miller, W. R. Sutton and P. Werbos, 67 - 96.
